# Efficient Learning Equilibrium *

**Ronen I. Brafman**
Computer Science Department
Ben-Gurion University
Beer-Sheva, Israel
email: brafman@cs.bgu.ac.il

**Moshe Tennenholtz**
Computer Science Department
Stanford University
Stanford, CA 94305
e-mail: moshe@robotics.stanford.edu

## Abstract

We introduce *efficient learning equilibrium* (ELE), a normative approach to learning in non cooperative settings. In ELE, the learning algorithms themselves are required to be in equilibrium. In addition, the learning algorithms arrive at a desired value after polynomial time, and deviations from a prescribed ELE become irrational after polynomial time. We prove the existence of an ELE in the perfect monitoring setting, where the desired value is the expected payoff in a Nash equilibrium. We also show that an ELE does not always exist in the imperfect monitoring case. Yet, it exists in the special case of common-interest games. Finally, we extend our results to general stochastic games.

## 1 Introduction

Reinforcement learning in the context of multi-agent interaction has attracted the attention of researchers in cognitive psychology, experimental economics, machine learning, artificial intelligence, and related fields for quite some time [8, 4]. Much of this work uses repeated games [3, 5] and stochastic games [10, 9, 7, 1] as models of such interactions.

The literature on learning in games in game theory [5] is mainly concerned with the understanding of learning procedures that **if** adopted by the different agents will converge at end to an equilibrium of the corresponding game. The game itself may be known; the idea is to show that simple dynamics lead to rational behavior, as prescribed by a Nash equilibrium. The learning algorithms themselves are not required to satisfy any rationality requirement; it is what they converge to, **if** adopted by **all** agents that should be in equilibrium. This is quite different from the classical perspective on learning in Artificial Intelligence, where the main motivation

for learning stems from the fact that the model of the environment is unknown. For example, consider a Markov Decision Process (MDP). If the rewards and transition probabilities are known then one can find an optimal policy using dynamic programming. The major motivation for learning in this context stems from the fact that the model (i.e. rewards and transition probabilities) is initially unknown. When facing uncertainty about the game that is played, game-theorists appeal to a Bayesian approach, which is completely different from a learning approach; the typical assumption in that approach is that there exists a probability distribution on the possible games, which is common-knowledge. The notion of equilibrium is extended to this context of games with incomplete information, and is treated as the appropriate solution concept. In this context, agents are assumed to be rational agents adopting the corresponding (Bayes-) Nash equilibrium, and learning is not an issue.

In this work we present an approach to learning in games, where there is no known distribution on the possible games that may be played – an approach that appears to be much more reflective of the setting studied in machine learning and AI and in the spirit of work on on-line algorithms in computer science. Adopting the framework of repeated games, we consider a situation where the learning algorithm is a strategy for an agent in a repeated game. This strategy takes an action at each stage based on its previous observations, and initially has no information about the identity of the game being played. Given the above, the following are natural requirements for the learning algorithms provided to the agents:

1. *Individual Rationality:* The learning algorithms themselves should be in equilibrium. It should be irrational for each agent to deviate from its learning algorithm, as long as the other agents stick to their algorithms, *regardless* of the what the actual game is.

2. *Efficiency:*

   (a) A deviation from the learning algorithm by a single agent (while the other stick to their algorithms) will become irrational (i.e. will lead to a situation where the deviator's payoff is not improved) after polynomially many stages.

   (b) If all agents stick to their prescribed learning algorithms then the expected payoff obtained by each agent within a polynomial number of steps will be (close to) the value it could have obtained in a Nash equilibrium, had the agents known the game from the outset.

A tuple of learning algorithms satisfying the above properties for a given *class* of games is said to be an *Efficient Learning Equilibrium*[ELE]. Notice that the learning algorithms should satisfy the desired properties for *every* game in a given class despite the fact that the actual game played is initially unknown. Such assumptions are typical to work in machine learning. What we borrow from the game theory literature is the criterion for rational behavior in multi-agent systems. That is, we take individual rationality to be associated with the notion of equilibrium. We also take the equilibrium of the actual (initially unknown) game to be our benchmark for success; we wish to obtain a corresponding value although we initially do not know which game is played.

In the remaining sections we formalize the notion of efficient learning equilibrium, and present it in a self-contained fashion. We also prove the existence of an ELE (satisfying all of the above desired properties) for a general class of games (repeated games with perfect monitoring) , and show that it does not exist for another. Our results on ELE can be generalized to the context of Pareto-ELE (where we wish to obtain maximal social surplus), and to general stochastic games. These will be mentioned only very briefly, due to space limitations. The discussion of these and other issues, as well as proofs of theorems, can be found in the full paper [2].

Technically speaking, the results we prove rely on a novel combination of the so-called folk theorems in economics, and a novel efficient algorithm for the punishment of deviators (in games which are initially unknown).

## 2   ELE: Definition

In this section we develop a definition of efficient learning equilibrium. For ease of exposition, our discussion will center on two-player repeated games in which the agents have an identical set of actions $A$. The generalization to $n$-player repeated games with different action sets is immediate, but requires a little more notation. The extension to stochastic games is fully discussed in the full paper [2].

A *game* is a model of multi-agent interaction. In a game, we have a set of players, each of whom performs some action from a given set of actions. As a result of the players' combined choices, some outcome is obtained which is described numerically in the form of a payoff vector, i.e., a vector of values, one for each of the players.

A common description of a (two-player) game is as a matrix. This is called a game in *strategic form*. The rows of the matrix correspond to player 1's actions and the columns correspond to player 2's actions. The entry in row $i$ and column $j$ in the game matrix contains the rewards obtained by the players if player 1 plays his $i^{th}$ action and player 2 plays his $j^{th}$ action.

In a *repeated game* (RG) the players play a given game $G$ repeatedly. We can view a repeated game, with respect to a game $G$, as consisting of infinite number of iterations, at each of which the players have to select an action of the game $G$. After playing each iteration, the players receive the appropriate payoffs, as dictated by that game's matrix, and move to a new iteration.

For ease of exposition we normalize both players' payoffs in the game $G$ to be non-negative reals between 0 and some positive constant $R_{max}$. We denote this interval (or set) of possible payoffs by $P = [0, R_{max}]$.

In a *perfect monitoring* setting, the set of possible histories of length $t$ is $(A^2 \times P^2)^t$, and the set of possible histories, $H$, is the union of the sets of possible histories for all $t \geq 0$, where $(A^2 \times P^2)^0$ is the empty history. Namely, the history at time $t$ consists of the history of actions that have been carried out so far, and the corresponding payoffs obtained by the players. Hence, in a perfect monitoring setting, a player can observe the actions selected and the payoffs obtained in the past, but does not know the game matrix to start with. In an *imperfect monitoring* setup, all that a player can observe following the performance of its action is the payoff it obtained and the action selected by the other player. The player cannot observe the other player's payoff. The definition of the possible histories for an agent naturally follows.

Finally, in a *strict imperfect monitoring* setting, the agent cannot observe the other agents' payoff or their actions.

Given an RG, a *policy* for a player is a mapping from $H$, the set of possible histories, to the set of possible probability distributions over $A$. Hence, a policy determines the probability of choosing each particular action for each possible history. A learning algorithm can be viewed as an instance of a policy.

We define the *value* for player 1 (resp. 2) of a policy profile $(\pi, \rho)$, where $\pi$ is a policy for player 1 and $\rho$ is a policy for player 2, using the *expected average reward criterion* as follows. Given an RG $M$ and a natural number $T$, we denote the expected $T$-step undiscounted average reward of player 1 (resp. 2) when the players follow the policy profile $(\pi, \rho)$, by $U_1(M, \pi, \rho, T)$ (resp. $U_2(M, \pi, \rho, T)$). We define $U_i(M, \pi, \rho) = \liminf_{T \to \infty} U_i(M, \pi, \rho, T)$ for $i = 1, 2$.

Let $\mathcal{M}$ denote a class of repeated games. A policy profile $(\pi, \rho)$ is a *learning equilibrium* w.r.t. $\mathcal{M}$ if $\forall \pi', \rho', M \in \mathcal{M}$, we have that $U_1(M, \pi', \rho) \leq U_1(M, \pi, \rho)$, and $U_2(M, \pi, \rho') \leq U_2(M, \pi, \rho)$. In this paper we mainly treat the class $\mathcal{M}$ of *all* repeated games with some fixed action profile (i.e., in which the set of actions available to all agents is fixed). However, in Section 4 we consider the class of common-interest repeated games. We shall stick to the assumption that both agents have a fixed, identical set $A$ of $k$ actions.

Our first requirement, then, is that learning algorithms will be treated as strategies. In order to be individually rational they should be the best response for one another. Our second requirement is that they rapidly obtain a desired value. The definition of this desired value may be a parameter, the most natural candidate – though not the only candidate – being the expected payoffs in a Nash equilibrium of the game. Another appealing alternative will be discussed later.

Formally, let $G$ be a (one-shot) game, let $M$ be the corresponding repeated game, and let $n(G)$ be a Nash-equilibrium of $G$. Then, denote the expected payoff of agent $i$ in $n(G)$ by $NV_i(n(G))$.

A policy profile $(\pi, \rho)$ is an *efficient learning equilibrium* with respect to the class of games $\mathcal{M}$ if for every $\epsilon > 0, 0 < \delta < 1$, there exists some $T > 0$, where $T$ is polynomial in $\frac{1}{\epsilon}, \frac{1}{\delta}$, and $k$, such that with probability of at least $1 - \delta$: (1) For every $t \geq T$ and for every repeated game $M \in \mathcal{M}$ (and its corresponding one-shot game, $G$), $U_i(M, \pi, \rho, t) \geq NV_i(n(G)) - \epsilon$ for $i = 1, 2$, for some Nash equilibrium $n(G)$, and (2) If player 1 (resp. 2) deviates from $\pi$ to $\pi'$ (resp. from $\rho$ to $\rho'$) in iteration $l$, then $U_1(M, \pi', \rho, l+t) \leq U_1(M, \pi, \rho, l+t) + \epsilon$ (resp. $U_2(M, \pi, \rho', l+t) \leq U_2(M, \pi, \rho, l+t) + \epsilon$) for every $t \geq T$.

Notice that a deviation is considered irrational if it does not increase the expected payoff by more than $\epsilon$. This is in the spirit of $\epsilon$-equilibrium in game theory. This is done mainly for ease of mathematical exposition. One can replace this part of the definition, while getting similar results, with the requirement of "standard" equilibrium, where a deviation will not improve the expected payoff, and even with the notion of strict equilibrium, where a deviation will lead to a decreased payoff. This will require, however, that we restrict our attention to games where there exist a Nash equilibrium in which the agents' expected payoffs are higher than their probabilistic maximin values.

The definition of ELE captures the insight of a normative approach to learning in non-cooperative settings. We assume that initially the game is unknown, but the agents will have learning algorithms that will rapidly lead to the values the players would have obtained in a Nash equilibrium had they known the game. Moreover, as mentioned earlier, the learning algorithms themselves should be in equilibrium. Notice that each agent's behavior should be the best response against the other agents' behaviors, and deviations should be irrational, regardless of what the actual (one-shot) game is.

## 3  Efficient Learning Equilibrium: Existence

Let $M$ be a repeated game in which $G$ is played at each iteration. Let $A = \{a_1, \ldots, a_k\}$ be the set of possible actions for both agents. Finally let there be an agreed upon ordering over the actions. The basic idea behind the algorithm is as follows. The agents collaborate in exploring the game. This requires $k^2$ moves. Next, each agent computes a Nash equilibrium of the game and follows it. If more than one equilibrium exists, then the first one according to the natural lexicographic ordering is used.[1] If one of the agents does not collaborate in the initial exploration phase, the other agent "punishes" this agent. We will show that efficient punishment is feasible. Otherwise, the agents have chosen a Nash-equilibrium, and it is irrational for them to deviate from this equilibrium unilaterally.

This idea combines the so-called folk-theorems in economics [6], and a technique for learning in zero-sum games introduced in [1]. Folk-theorems in economics deal with a technique for obtaining some desired behavior by making a threat of employing a punishing strategy against a deviator from that behavior. When both agents are equipped with corresponding punishing strategies, the desired behavior will be obtained in equilibrium (and the threat will not be materialized – as a deviation becomes irrational). In our context however, when an agent deviates in the exploration phase, then the game is not fully known, and hence punishment is problematic; moreover, we wish the punishment strategy to be an efficient algorithm (both computationally, and in the time a punishment will materialize and make deviations irrational). These are addressed by having an efficient punishment algorithm that guarantees that the other agent will not obtain more than its maximin value, after polynomial time, although the game is initially unknown to the punishing agent. The latter is based on the ideas of our R-max algorithm, introduced in [1].

More precisely, consider the following algorithm, termed the ELE algorithm.

**The ELE algorithm:**

Player 1 performs action $a_i$ one time after the other for $k$ times, for all $i = 1, 2, ..., k$. In parallel, player 2 performs the sequence of actions $(a_1, \ldots, a_k)$ $k$ times.

If both players behaved according to the above then a Nash equilibrium of the corresponding (revealed) game is computed, and the players behave according to the corresponding strategies from that point on. If several Nash equilibria exist, one is selected based on a pre-determined lexicographic ordering.

If one of the players deviated from the above, we shall call this player *the adversary* and the other player *the agent*. Let $G$ be the $R_{max}$-sum game in which the adversary's payoff is identical to his payoff in the original game, and where the agent's payoff is $R_{max}$ minus the adversary payoffs. Let $M$ denote the corresponding repeated game. Thus, $G$ is a constant-sum game where the agent's goal is to minimize the adversary's payoff. Notice that some of these payoffs will be unknown (because the adversary did not cooperate in the exploration phase). The agent now plays according to the following algorithm:

**Initialize:** Construct the following model $M'$ of the repeated game $M$, where the game $G$ is replaced by a game $G'$ where all the entries in the game matrix are assigned the rewards $(R_{max}, 0)$.[2]

In addition, we associate a boolean valued variable with each joint-action {*assumed, known*}. This variable is initialized to the value *assumed*.

**Repeat:**

**Compute and Act:** Compute the optimal probabilistic maximin of $G'$ and execute it.

**Observe and update:** Following each joint action do as follows: Let $a$ be the action the agent performed and let $a'$ be the adversary's action. If $(a, a')$ is performed for the first time, update the reward associated with $(a, a')$ in $G'$, as observed, and mark it *known*. Recall – the agent takes its payoff to be complementary to the (observed) adversary's payoff.

We can show that the policy profile in which both agents use the ELE algorithm is indeed an ELE. Thus:

**Theorem 1** *Let $\mathcal{M}$ be a class of repeated games. Then, there exists an ELE w.r.t. $\mathcal{M}$ given perfect monitoring.*

The proof of the above Theorem, contained in the full paper, is non-trivial. It rests on the ability of the agent to "punish" the adversary quickly, making it irrational for the adversary to deviate from the ELE algorithm.

## 4 Imperfect monitoring

In the previous section we discussed the existence of an ELE in the context of the perfect monitoring setup. This result allows us to show that our concepts provide not only a normative, but also a constructive approach to learning in general non-cooperative environments. An interesting question is whether one can go beyond that and show the existence of an ELE in the imperfect monitoring case as well. Unfortunately, when considering the class $\mathcal{M}$ of all games, this is not possible.

**Theorem 2** *There exist classes of games for which an ELE does not exist given imperfect monitoring.*

**Proof (sketch):** We will consider the class of all $2 \times 2$ games and we will show that an ELE does not exist for this class under imperfect monitoring.

Consider the following games:

1. G1:

$$M = \begin{pmatrix} 6, & 0 & 0, \, 100 \\ 5, & -100 & 1, \, 500 \end{pmatrix}$$

2. G2:

$$M = \begin{pmatrix} 6, \, 9 & 0, \, 1 \\ 5, \, 11 & 1, \, 10 \end{pmatrix}$$

Notice that the payoffs obtained for a joint action in G1 and G2 are identical for player 1 and are different for player 2.

The only equilibrium of G1 is where both players play the second action, leading to (1,500). The only equilibrium of G2 is where both players play the first action, leading to (6,9). (These are unique equilibria since they are obtained by removal of strictly dominated strategies.)

Now, assume that an ELE exists, and look at the corresponding policies of the players in that equilibrium. Notice that in order to have an ELE, we must visit the entry (6,9) most of the times if the game is G2 and visit the entry (1,500) most of the times if the game is G1; otherwise, player 1 (resp. player 2) will not obtain a high enough value in G2 (resp. G1), since its other payoffs in G2 (resp. G1) are lower than that.

Given the above, it is rational for player 2 to deviate and pretend that the game is always G1 and behave according to what the suggested equilibrium policy tells it to do in that case. Since the game might be actually G1, and player 1 cannot tell the difference, player 2 will be able to lead to playing the second action by both players for most times also when the game is G2, increasing its payoff from 9 to 10, contradicting ELE.   ■

The above result demonstrates that without additional assumptions, one cannot provide an ELE under imperfect monitoring. However, for certain restricted classes of games, we can provide an ELE under imperfect monitoring, as we now show.

A game is called a *common-interest* game if for every joint-action, all agents receive the same reward. We can show:

**Theorem 3** *Let $\mathcal{M}_{c-i}$ be the class of common-interest repeated games in which the number of actions each agent has is $a$. There exists an ELE for $\mathcal{M}_{c-i}$ under* strict *imperfect monitoring.*

**Proof (sketch):** The agents use the following algorithm: for $m$ rounds, each agent randomly selects an action. Following this, each agent plays the action that yielded the best reward. If multiple actions led to the best reward, the one that was used first is selected. $m$ is selected so that with probability $1 - \delta$ every joint-action will be selected. Using Chernoff bound we can choose $m$ that is polynomial in the size of the game (which is $a^k$, where $k$ is the number of agents) and in $1/\delta$.   ■

This result improves previous results in this area, such as the combination of Q-learning and fictitious play used in [3]. Not only does it provably converge in polynomial time, it is also guaranteed, with probability of $1 - \delta$ to converge to the optimal Nash-equilibrium of the game rather than to an arbitrary (and possibly non-optimal) Nash-equilibrium.

## 5 Conclusion

We defined the concept of an efficient learning equilibria – a normative criterion for learning algorithms. We showed that given perfect monitoring a learning algorithm satisfying ELE exists, while this is not the case under imperfect monitoring. In the full paper [2] we discuss related solution concepts, such as Pareto ELE. A Pareto ELE is similar to a (Nash) ELE, except that the requirement of attaining the expected payoffs of a Nash equilibrium is replaced by that of maximizing social surplus. We show that there fexists a Pareto-ELE for any perfect monitoring setting, and that a Pareto ELE does not always exist in an imperfect monitoring setting. In the full paper we also extend our discussion from repeated games to infinite horizon stochastic games under the average reward criterion. We show that under perfect monitoring, there always exists a Pareto ELE in this setting. Please refer to [2] for additional details and the full proofs.

## Footnotes

*The second author permanent address is: Faculty of Industrial Engineering and Management, Technion, Haifa 32000, Israel. This work was supported in part by Israel Science Foundation under Grant #91/02-1. The first author is partially supported by the Paul Ivanier Center for Robotics and Production Management.

[1]In particular, the agents can choose the equilibrium selected by a fixed shared algorithm.

[2]The value 0 given to the adversary does not play an important role here.

## References

[1] R. I. Brafman and M. Tennenholtz. R-max – a general polynomial time algorithm for near-optimal reinforcement learning. In *IJCAI'01*, 2001.

[2] R. I. Brafman and M. Tennenholtz. Efficient learning equilibrium. Technical Report 02-06, Dept. of Computer Science, Ben-Gurion University, 2002.

[3] C. Claus and C. Boutilier. The dynamics of reinforcement learning in cooperative multi-agent systems. In *Proc. Workshop on Multi-Agent Learning*, pages 602–608, 1997.

[4] I. Erev and A.E. Roth. Predicting how people play games: Reinforcement learning in games with unique strategy equilibrium. *American Economic Review*, 88:848–881, 1998.

[5] D. Fudenberg and D. Levine. *The theory of learning in games*. MIT Press, 1998.

[6] D. Fudenberg and J. Tirole. *Game Theory*. MIT Press, 1991.

[7] J. Hu and M.P. Wellman. Multi-agent reinforcement learning: Theoretical framework and an algorithms. In *Proc. 15th ICML*, 1998.

[8] L. P. Kaelbling, M. L. Littman, and A. W. Moore. Reinforcement learning: A survey. *Journal of AI Research*, 4:237–285, 1996.

[9] M. L. Littman. Markov games as a framework for multi-agent reinforcement learning. In *Proc. 11th ICML*, pages 157–163, 1994.

[10] L.S. Shapley. Stochastic Games. In *Proc. Nat. Acad. Scie. USA*, volume 39, pages 1095–1100, 1953.
